# Programmable Reinforcement Learning Agents

**David Andre and Stuart J. Russell**
Computer Science Division, UC Berkeley, CA 94702
{dandre,russell}@cs.berkeley.edu

## Abstract

We present an expressive agent design language for reinforcement learning that allows the user to constrain the policies considered by the learning process.The language includes standard features such as parameterized subroutines, temporary interrupts, aborts, and memory variables, but also allows for *unspecified* choices in the agent program. For learning that which isn't specified, we present provably convergent learning algorithms. We demonstrate by example that agent programs written in the language are concise as well as modular. This facilitates state abstraction and the transferability of learned skills.

## 1 Introduction

The field of reinforcement learning has recently adopted the idea that the application of prior knowledge may allow much faster learning and may indeed be essential if real-world environments are to be addressed. For learning behaviors, the most obvious form of prior knowledge provides a *partial description of desired behaviors*. Several languages for partial descriptions have been proposed, including Hierarchical Abstract Machines (HAMs) [8], semi-Markov options [12], and the MAXQ framework [4].

This paper describes extensions to the HAM language that substantially increase its expressive power, using constructs borrowed from programming languages. Obviously, increasing expressiveness makes it easier for the user to supply whatever prior knowledge is available, and to do so more concisely. (Consider, for example, the difference between wiring up Boolean circuits and writing Java programs.) More importantly, the availability of an expressive language allows the agent to learn and generalize behavioral abstractions that would be far more difficult to learn in a less expressive language. For example, the ability to specify *parameterized* behaviors allows multiple behaviors such as $WalkEast$, $WalkNorth$, $WalkWest$, $WalkSouth$ to be combined into a single behavior $Walk(d)$ where $d$ is a direction parameter. Furthermore, if a behavior is appropriately parameterized, decisions within the behavior can be made independently of the "calling context" (the hierarchy of tasks within which the behavior is being executed). This is crucial in allowing behaviors to be learned and reused as general skills.

Our extended language includes parameters, interrupts, aborts (i.e., interrupts without resumption), and local state variables. Interrupts and aborts in particular are very important in physical behaviors—more so than in computation—and are crucial in allowing for modularity in behavioral descriptions. These features are all common in robot programming languages [2, 3, 5]; the key element of our approach is that behaviors need only be *partially* described; reinforcement learning does the rest.

To tie our extended language to existing reinforcement learning algorithms, we utilize Parr and Russell's [8] notion of the joint semi-Markov decision process (SMDP) created when

a HAM is composed with an environment (modeled as an MDP). The joint SMDP state space consists of the cross-product of the machine states in the HAM and the states in the original MDP; the dynamics are created by the application of the HAM in the MDP. Parr and Russell showed that an optimal solution to the joint SMDP is both learnable and yields an optimal solution to the original MDP *in the class of policies expressed by the HAM* (so-called *hierarchical optimality*). Furthermore, Parr and Russell show that the joint SMDP can be reduced to an equivalent SMDP with a state space consisting only of the states where the HAM does not specify an action, which reduces the complexity of the SMDP problem that must be solved. We show that these results hold for our extended language of Programmable HAMs (PHAMs).

To demonstrate the usefulness of the new language, we show a small, complete program for a complex environment that would require a much larger program in previous formalisms. We also show experimental results verifying the convergence of the learning process for our language.

## 2  Background

An MDP is a 4-tuple, $(\mathcal{S}, \mathcal{A}, \mathcal{T}, \mathcal{R})$, where $\mathcal{S}$ is a set of states, $\mathcal{A}$ is a set of actions, $\mathcal{T}$ is a probabilistic transition function mapping $\mathcal{S} \times \mathcal{A} \times \mathcal{S} \rightarrow [0, 1]$, and $\mathcal{R}$ is a reward function mapping $\mathcal{S} \times \mathcal{A} \times \mathcal{S}$ to the reals. In this paper, we focus on infinite-horizon MDPs with a discount factor $\beta$. A solution to a MDP is an optimal policy $\pi^*$ that maps from $\mathcal{S} \rightarrow \mathcal{A}$ and achieves maximum expected discounted reward for the agent. An SMDP (semi-Markov decision process) allows for actions that take more than one time step. $\mathcal{T}$ is modified to be a mapping from $\mathcal{S}, \mathcal{A}, \mathcal{S}, \mathbf{N} \rightarrow [0, 1]$, where $\mathbf{N}$ is the natural numbers; i.e., it specifies a distribution over both output states and action durations. $\mathcal{R}$ is then a mapping from $\mathcal{S}, \mathcal{A}, \mathcal{S}, \mathbf{N}$ to the reals. The discount factor, $\beta$, is generalized to be a function, $\beta(s, a)$, that represents the expected discount factor when action $a$ is taken in state $s$. Our definitions follow those common in the literature [9, 6, 4].

The HAM language [8] provides for partial specification of agent programs. A HAM program consists of a set of partially specified Moore machines. Transitions in each machine may depend stochastically on (features of) the environment state, and the outputs of each machine are primitive actions or nonrecursive invocations of other machines. The states in each machine can be of four types: {*start, stop, action, choice*}. Each machine has a single distinguished *start* state and may have one or more distinguished *stop* states. When a machine is invoked, control starts at the *start* state; *stop* states return control back to the calling machine. An *action* state executes an action. A *call* state invokes another machine as a subroutine. A *choice* state may have several possible next states; after learning, the choice is reduced to a single next state.

## 3  Programmable HAMs

Consider the problem of creating a HAM program for the Deliver–Patrol domain presented in Figure 1, which has 38,400 states. In addition to delivering mail and picking up occasional additional rewards while patrolling (both of which require efficient navigation and safe maneuvering), the robot must keep its battery charged (lest it be stranded) and its camera lens clean (lest it crash). It must also decide whether to move quickly (incurring collision risk) or slowly (delaying reward), depending on circumstances.

Because all the 5×5 "rooms" are similar, one can write a "traverse the room" HAM routine that works in all rooms, but a different routine is needed for each *direction* (north–south, south–north, east–west, etc.). Such redundancy suggests the need for a "traverse the room" routine that is *parameterized* by the desired direction.

Consider also the fact that the robot should clean its camera lens *whenever it gets dirty*.

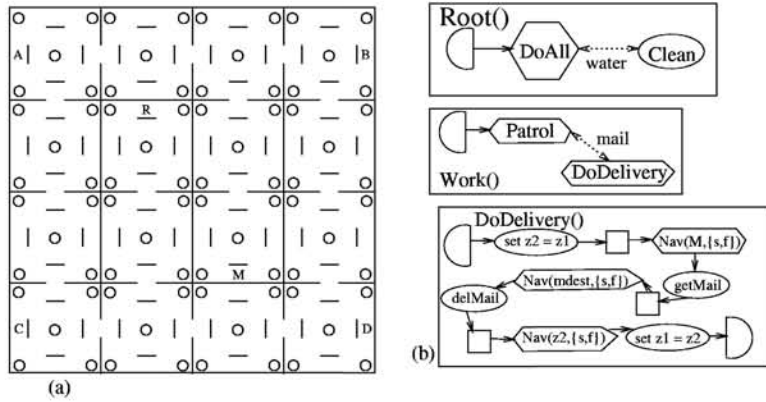

Figure 1: (a) The Deliver–Patrol world. Mail appears at $M$ and must be delivered to the appropriate location. Additional rewards appear sporadically at $A$, $B$, $C$, and $D$. The robot's battery may be recharged at $R$. The robot is penalized for colliding with walls and "furniture" (small circles). (b) Three of the PHAMs in the partial specification for the Deliver–Patrol world. Right-facing half-circles are start states, left-facing half-circles are stop states, hexagons are call states, ovals are primitive actions, and squares are choice points. $z1$ and $z2$ are memory variables. When arguments to call states are in braces, then the choice is over the arguments to pass to the subroutine. The $Root()$ PHAM specifies an interrupt to clean the camera lens whenever it gets dirty; the $Work()$ PHAM interrupts its patrolling whenever there is mail to be delivered.

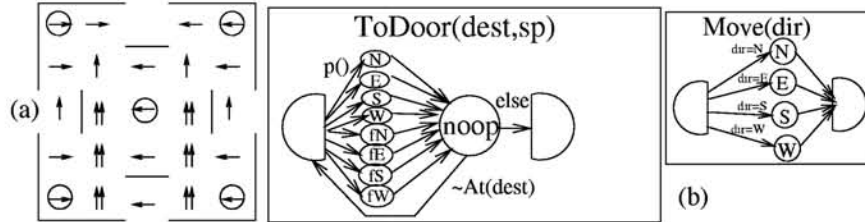

Figure 2: (a) A room in the Deliver–Patrol domain. The arrows in the drawing of the room indicate the behavior specified by the $p()$ transition function in *ToDoor(dest,sp)*. Two arrows indicate a "fast" move (*fN,fS,fE,fW*), whereas a single arrow indicates a slow move (*N, S, E, W*). (b) The *ToDoor(dest,sp)* and *Move(dir)* PHAMs.

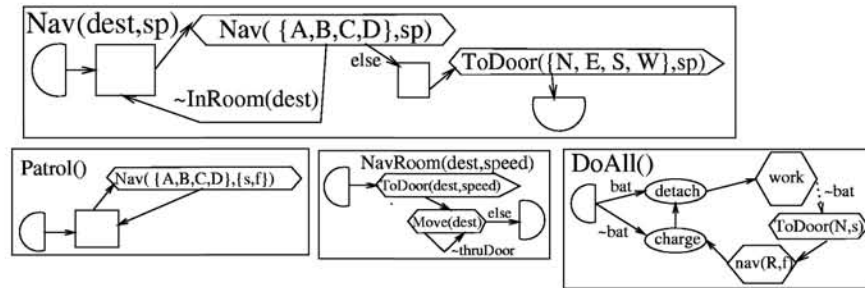

Figure 3: The remainder of the PHAMs for the Deliver–Patrol domain. *Nav(dest,sp)* leaves route choices to be learned through experience. Similarly, *Patrol()* does not specify the sequence of locations to check.

In the HAM language, this conditional action must be inserted *after every state in every HAM*. An interrupt mechanism with appropriate scoping would obviate the need for such widespread mutilation.

The PHAM language has these additional characteristics. We provide here an informal summary of the language features that enable concise agent programs to be written. The 9 PHAMs for the Deliver–Patrol domain are presented in Figure 1(b), Figure 2(b), and Figure 3. The corresponding HAM program requires 63 machines, many of which have significantly more states than their PHAM counterparts.

The PHAM language adds several structured programming constructs to the HAM language. To enable this, we introduce two additional types of states in the PHAM: *internal* states, which execute an internal computational action (such as setting memory variables to a function of the current state), and *null* states, which have no direct effect and are used for computational convenience.

**Parameterization** is key for expressing concise agent specifications, as can be seen in the Deliver–Patrol task. Subroutines take a number of parameters, $\theta_1, \theta_2, ...\theta_k$, the values of which must be filled in by the calling subroutine (and can depend on any function of the machine, parameter, memory, and environment state). In Figure 2(b), the subroutine *Move(dir)* is shown. The *dir* parameter is supplied by the *NavRoom* subroutine. The *ToDoor(dest,speed)* subroutine is for navigating a single room of the agent's building. The $p()$ is a transition function that stores a parameterized policy for getting to each door. The policy for $(N, f)$ (representing the North door, going fast) is shown in Figure 2(a). Note that by using parameters, the control for navigating a room is quite modular, and is written once, instead of once for each direction and speed.

**Aborts and interrupts** allow for modular agent specification. As well as the camera-lens interrupt described earlier, the robot needs to abort its current activity if the battery is low and should interrupt its patrolling activity if mail arrives for delivery. The PHAM language allows abort conditions to be specified at the point where a subroutine is invoked within a calling routine; those conditions are in force until the subroutine exits. For each abort condition, an "abort handler" state is specified within the calling routine, to which control returns if the condition becomes true. (For interrupts, normal execution is resumed once the handler completes.) Graphically, aborts are depicted as labelled dotted lines (e.g., in the *DoAll()* PHAM in Figure 3), and interrupts are shown as labelled dashed lines with arrows on both ends (e.g., in the *Work()* PHAM in Figure 1(b)).

**Memory variables** are a feature of nearly every programming language. Some previous research has been done on using memory variables in reinforcement learning in partially observable domains [10]. For an example of memory use in our language, examine the *DoDelivery* subroutine in Figure 1(b), where $z_2$ is set to another memory value (set in *Nav(dest,sp)*). $z_2$ is then passed as a variable to the *Nav* subroutine. Computational functions such as *dest* in the *Nav(dest,sp)* subroutine are restricted to be recursive functions taking effectively zero time. A PHAM is assumed to have a finite number of memory variables, $z_1, \ldots, z_n$, which can be combined to yield the memory state, $Z$. Each memory variable has finite domain $\mathcal{D}(z_i)$. The agent can set memory variables by using *internal* states, which are computational action states with actions in the following format: (set $z_1\ \psi(m, \theta, x, Z)$), where $\psi(m, \theta, x, Z)$ is a function taking the machine, parameter, environment, and memory state as parameters. The transition function, parameter-setting functions, and choice functions take the memory state into account as well.

## 4    Theoretical Results

Our results mirror those obtained in [9]. In summary (see also Figure 4): The composition $\mathcal{H} \circ \mathcal{M}$ of a PHAM $\mathcal{H}$ with the underlying MDP $\mathcal{M}$ is defined using the cross product of states in $\mathcal{H}$ and $\mathcal{M}$. This composition is in fact an SMDP. Furthermore, solutions to $\mathcal{H} \circ \mathcal{M}$

yield optimal policies for the original MDP, among those policies expressed by the PHAM. Finally, $\mathcal{H} \circ \mathcal{M}$ may be *reduced* to an equivalent SMDP whose states are just the *choice points*, i.e., the joint states where the machine state is a choice state. See [1] for the proofs.

**Definition 1 (Programmable Hierarchical Abstract Machines: PHAMs)** *A PHAM is a tuple* $\mathcal{H} = (\mu, \Theta, \delta, \rho, \xi, \mathcal{I}, \mu_{\mathcal{I}}, \mathcal{A}, \mu_{\mathcal{A}}, \mathcal{Z}, \Psi)$, *where* $\mu$ *is the set of machine states in* $\mathcal{H}$, $\Theta$ *is the space of possible parameter settings,* $\delta$ *is the transition function, mapping* $\mu \times \Theta \times \mathcal{Z} \times X \times \mu$ *to* $[0, 1]$, $\rho$ *is a mapping from* $\mu \times \Theta \times \mathcal{Z} \times X \times \Theta$ *to* $[0, 1]$ *and expresses the parameter choice function,* $\xi$ *maps from* $\mu \times \Theta \times \mathcal{Z} \times X$ *to subsets of* $\mu$ *and expresses the allowed choices at choice states,* $\mathcal{I}(m)$ *returns the interrupt condition at a call state,* $\mu_{\mathcal{I}}(m)$ *specifies the handler of an interrupt,* $\mathcal{A}(m)$ *returns the abort condition at a call state,* $\mu_{\mathcal{A}}(m)$ *specifies the handler of an abort,* $\mathcal{Z}$ *is the set of possible memory configurations, and* $\Psi(m)$ *is a complex function expressing which computational internal function is used at* internal *states, and to which memory variable the result is assigned.*

**Theorem 1** *For any MDP,* $\mathcal{M}$ *and any PHAM,* $\mathcal{H}$, *the operation of* $\mathcal{H}$ *in* $\mathcal{M}$ *induces a joint SMDP, called* $\mathcal{H} \circ \mathcal{M}$. *If* $\pi$ *is an optimal solution for* $\mathcal{H} \circ \mathcal{M}$, *then the primitive actions specified by* $\pi$ *constitute an optimal policy for* $\mathcal{M}$ *among those consistent with* $\mathcal{H}$.

The state space of $\mathcal{H} \circ \mathcal{M}$ may be enormous. As is illustrated in Figure 4, however, we can obtain significant further savings, just as in [9]. First, not all pairs of PHAM and MDP states will be reachable from the initial state; second, the complexity of the induced SMDP is solely determined by the number of reachable choice points.

**Theorem 2** *For any MDP* $\mathcal{M}$ *and PHAM* $\mathcal{H}$, *let* $\mathcal{C}$ *be the set of choice points in* $\mathcal{H} \circ \mathcal{M}$. *There exists an SMDP called* reduce($\mathcal{H} \circ \mathcal{M}$) *with states* $\mathcal{C}$ *such that the optimal policy for* reduce($\mathcal{H} \circ \mathcal{M}$) *corresponds to an optimal policy for* $\mathcal{M}$ *among those consistent with* $\mathcal{H}$.

The reduced SMDP can be solved by offline, model-based techniques using the method given in [9] for constructing the reduced model. Alternatively, and much more simply, we can solve it using online, model-free HAMQ-learning [8], which learns directly in the reduced state space of choice points. Starting from a choice state $\omega$ where the agent takes action $a$, the agent keeps track of the reward $r_{tot}$ and discount $\beta_{tot}$ accumulated on the way to the next choice point, $\omega'$. On each step, the agent encounters reward $r_i$ and discount $\beta_i$ (note that $\beta_i$ is 0 exactly when the agent transitions only in the PHAM and not in the MDP), and updates the totals as follows:

$$r_{tot} \leftarrow r_{tot} + \beta_{tot} r_i; \ \beta_{tot} \leftarrow \beta_{tot} \beta_i \ .$$

The agent maintains a Q-table, $Q(\omega, a)$, indexed by choice state and action. When the agent gets to the next choice state, $w'$, it updates the Q-table as follows:

$$Q(\omega, a) \leftarrow (1 - \alpha)Q(\omega, a) + \alpha[r_{tot} + \beta_{tot} \max_u Q(\omega', u)] \ .$$

We have the following theorem.

**Theorem 3** *For a PHAM* $\mathcal{H}$ *and and MDP* $\mathcal{M}$, *HAMQ-learning will converge to an optimal policy for* reduce($\mathcal{H} \circ \mathcal{M}$), *with probability 1, with appropriate restrictions on the learning rate.*

## 5 Expressiveness of the PHAM language

As shown by Parr [9], the HAM language is at least as expressive as some existing action languages including options [12] and full-$\beta$ models [11]. The PHAM language is substantially more expressive than HAMs. As mentioned earlier, the Deliver–Patrol PHAM program has 9 machines whereas the HAM program requires 63. In general, the additional number of states required to express a PHAM as a pure HAM is $|\mathcal{D}(Z) \times \mathcal{C} \times \Theta|$, where $\mathcal{D}(Z)$ is the memory state space, $\mathcal{C}$ is the set of possible abort/interrupt contexts, and $\Theta$ is the total parameter space. We also developed a PHAM program for the 3,700-state maze world used by Parr and Russell [8]. The HAM used in their experiments had 37 machines; the PHAM program requires only 7.

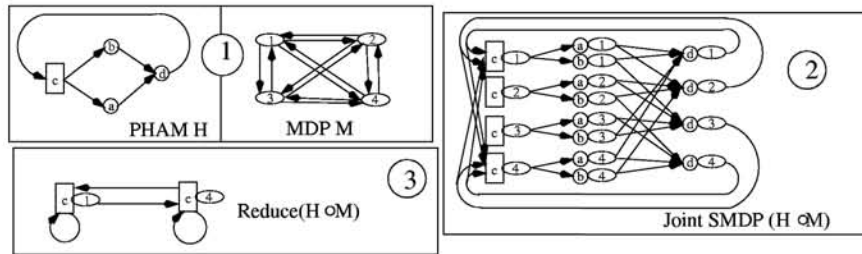

Figure 4: A schematic illustration of the formal results. (1) The top two diagrams are of a PHAM fragment with 1 choice state and 3 action states (of which one, labelled $d$, is the start state). The MDP has 4 states, and action $d$ always leads to state 1 or 4. The composition, $\mathcal{H} \circ \mathcal{M}$, is shown in (2). Note that there are no incoming arcs to the states $< c, 2 >$ or $< c, 3 >$. In (3), *reduce($\mathcal{H} \circ \mathcal{M}$)* is shown. There are only 2 states in the reduced SMDP because there are no incoming arcs to the states $< c, 2 >$ or $< c, 3 >$.

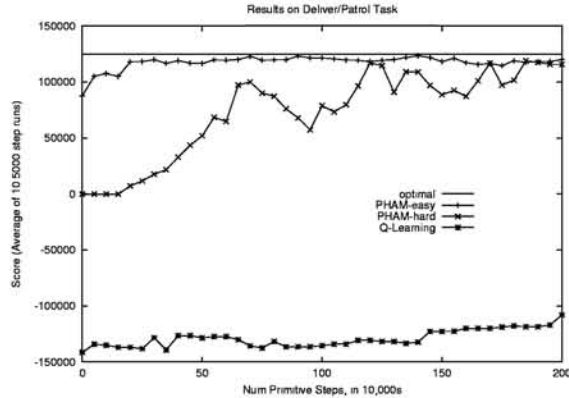

Figure 5: Learning curves for the Deliver/Patrol domain, averaged over 25 runs. X-axis: number of primitive steps. Y-axis: value of the policy measured by ten 5,000 step trials. PHAM-hard refers to the PHAMs given in this paper. PHAM-easy refers to a more complete PHAM, leaving unspecified only the speed of travel for each activity.

With respect to the induced choice points, the Deliver–Patrol PHAM induces 7,816 choice points in the joint SMDP, compared with 38,400 in the original MDP. Furthermore, only 15,800 Q-values must be learned, compared with 307,200 for flat Q-learning. Figure 5 shows empirical results for the Deliver–Patrol problem, indicating that Q-learning with a suitable PHAM program is far faster than flat Q-learning. (Parr and Russell observed similar results for the maze world, where HAMQ-learning finds a good policy in 270,000 iterations compared to 9,000,000 for flat Q-learning.) Note that equivalent HAM and PHAM programs yield identical reductions in the number of choice points and identical speedups in Q-learning. Thus, one might argue that PHAMs do not offer any advantage over HAMs, as they can express the same set of behaviors. However, this would be akin to arguing that the Java programming language offers nothing over Boolean circuits. Ease of expression and the ability to utilize greater modularity can greatly ease the task of coding reinforcement learning agents that take advantage of prior knowledge.

An interesting feature of PHAMs was observed in the Deliver–Patrol domain. The initial PHAM program was constructed on the assumption that the agent should patrol among $A$, $B$, $C$, $D$ unless there is mail to be delivered. However, the specific rewards are such that the optimal behavior is to loiter in the mail room until mail arrives, thereby avoiding costly

delays in mail delivery. The PHAM-Q learning agents learned this optimal behavior by "retargeting" the *Nav* routine to stay in the mail room rather than go to the specified destination. This example demonstrates the difference between constraining behavior through structure and constraining behavior through subgoals: the former method may give the agent greater flexibility but may yield "surprising" results. In another experiment, we constrained the PHAM further to prevent loitering. As expected, the agent learned a suboptimal policy in which *Nav* had the intended meaning of travelling to a specified destination. This experience suggests a natural debugging cycle in which the agent designer may examine learned behaviors and adjust the PHAM program accordingly.

The additional features of the PHAM language allow direct expression of programs from other formalisms that are not easily expressed using HAMs. For example, programs in Dietterich's MAXQ language [4] are written easily as PHAMs, but not as HAMs because the MAXQ language allows parameters. The language of teleo-reactive (TR) programs [7, 2] relies on a *prioritized* set of condition–action rules to achieve a goal. Each action can itself be another TR program. The TR architecture can be implemented directly in PHAMs using the abort mechanism [1].

# 6 Future work

Our long-term goal in this project is to enable true cross-task learning of skilled behavior. This requires *state abstraction* in order to learn choices within PHAMs that are applicable in large classes of circumstances rather than just to each invocation instance separately. Dietterich [4] has derived conditions under which state abstraction can be done within his MAXQ framework without sacrificing recursive optimality (a weaker form of optimality than hierarchical optimality). We have developed a similar set of conditions, based on a new form of value function decomposition, such that PHAM learning maintains hierarchical optimality. This decomposition critically depends on the modularity of the programs introduced by the language extensions presented in this paper.

Recently, we have added recursion and complex data structures to the PHAM language, incorporating it into a standard programming language (Lisp). This provides the PHAM programmer with a very powerful set of tools for creating adaptive agents.

# References

[1] D. Andre. Programmable HAMs. www.cs.berkeley.edu/~dandre/pham.ps, 2000.

[2] S. Benson and N. Nilsson. Reacting, planning and learning in an autonomous agent. In K. Furukawa, D. Michie, and S. Muggleton, editors, *Machine Intelligence 14*. 1995.

[3] G. Berry and G. Gonthier. The Esterel synchronous programming language: Design, semantics, implementation. *Science of Computer Programming*, 19(2):87–152, 1992.

[4] T. G. Dietterich. State abstraction in MAXQ hierarchical RL. In *NIPS 12*, 2000.

[5] R.J. Firby. Modularity issues in reactive planning. In *AIPS 96*, pages 78–85. AAAI Press, 1996.

[6] L. P. Kaelbling, M. L. Littman, and A. W. Moore. Reinforcement learning: A survey. *JAIR*, 4:237–285, 1996.

[7] N. J. Nilsson. Teleo-reactive programs for agent control. *JAIR*, 1:139–158, 1994.

[8] R. Parr and S. J. Russell. Reinforcement learning with hierarchies of machines. In *NIPS 10*, 1998.

[9] R. Parr. *Hierarchical Control and Learning for MDPs*. PhD thesis, UC Berkeley, 1998.

[10] L. Peshkin, N. Meuleau, and L. Kaelbling. Learning policies with external memory. In *ICML*, 1999.

[11] R. Sutton. Temporal abstraction in reinforcement learning. In *ICML*, 1995.

[12] R. Sutton, D. Precup, and S. Singh. Between MDPs and semi-MDPs: A framework for temporal abstraction in reinforcement learning. *Artificial Intelligence*, 112(1):181–211, February 1999.
